# Improved Switching among Temporally Abstract Actions

**Richard S. Sutton   Satinder Singh**
AT&T Labs
Florham Park, NJ 07932
{sutton,baveja}@research.att.com

**Doina Precup   Balaraman Ravindran**
University of Massachusetts
Amherst, MA 01003-4610
{dprecup,ravi}@cs.umass.edu

## Abstract

In robotics and other control applications it is commonplace to have a pre-existing set of controllers for solving subtasks, perhaps hand-crafted or previously learned or planned, and still face a difficult problem of how to choose and switch among the controllers to solve an overall task as well as possible. In this paper we present a framework based on Markov decision processes and semi-Markov decision processes for phrasing this problem, a basic theorem regarding the improvement in performance that can be obtained by switching flexibly between given controllers, and example applications of the theorem. In particular, we show how an agent can plan with these high-level controllers and then use the results of such planning to find an even better plan, by modifying the existing controllers, with negligible additional cost and no re-planning. In one of our examples, the complexity of the problem is reduced from 24 billion state-action pairs to less than a million state-controller pairs.

In many applications, solutions to parts of a task are known, either because they were hand-crafted by people or because they were previously learned or planned. For example, in robotics applications, there may exist controllers for moving joints to positions, picking up objects, controlling eye movements, or navigating along hallways. More generally, an intelligent system may have available to it several temporally extended courses of action to choose from. In such cases, a key challenge is to take full advantage of the existing temporally extended actions, to choose or switch among them effectively, and to plan at their level rather than at the level of individual actions.

Recently, several researchers have begun to address these challenges within the framework of reinforcement learning and Markov decision processes (e.g., Singh, 1992; Kaelbling, 1993; Dayan & Hinton, 1993; Thrun and Schwartz, 1995; Sutton, 1995; Dietterich, 1998; Parr & Russell, 1998; McGovern, Sutton & Fagg, 1997). Common to much of this recent work is the modeling of a temporally extended action as a policy (controller) and a condition for terminating, which we together refer to as an *option* (Sutton, Precup & Singh, 1998). In this paper we consider the problem of effectively combining given options into one overall policy, generalizing prior work by Kaelbling (1993). Sections 1–3 introduce the framework; our new results are in Sections 4 and 5.

# 1   Reinforcement Learning (MDP) Framework

In a *Markov decision process (MDP)*, an agent interacts with an environment at some discrete, lowest-level time scale $t = 0, 1, 2, \ldots$ On each time step, the agent perceives the state of the environment, $s_t \in \mathcal{S}$, and on that basis chooses a *primitive* action, $a_t \in \mathcal{A}$. In response to each action, $a_t$, the environment produces one step later a numerical reward, $r_{t+1}$, and a next state, $s_{t+1}$. The *one-step model* of the environment consists of the one-step state-transition probabilities and the one-step expected rewards,

$$p_{ss'}^a = \Pr\{s_{t+1} = s' \mid s_t = s, a_t = a\} \qquad \text{and} \qquad r_s^a = E\{r_{t+1} \mid s_t = s, a_t = a\},$$

for all $s, s' \in \mathcal{S}$ and $a \in \mathcal{A}$. The agent's objective is to learn an *optimal Markov policy*, a mapping from states to probabilities of taking each available primitive action, $\pi : \mathcal{S} \times \mathcal{A} \to [0, 1]$, that maximizes the expected discounted future reward from each state $s$:

$$V^\pi(s) = E\left\{r_{t+1} + \gamma r_{t+2} + \cdots \mid s_t = s, \pi\right\} = \sum_{a \in \mathcal{A}_s} \pi(s, a)[r_s^a + \gamma \sum_{s'} p_{ss'}^a V^\pi(s')],$$

where $\pi(s, a)$ is the probability with which the policy $\pi$ chooses action $a \in \mathcal{A}_s$ in state $s$, and $\gamma \in [0, 1]$ is a *discount-rate* parameter. $V^\pi(s)$ is called the *value* of state $s$ under policy $\pi$, and $V^\pi$ is called the *state-value function* for $\pi$. The *optimal* state-value function gives the value of a state under an optimal policy: $V^*(s) = \max_\pi V^\pi(s) = \max_{a \in \mathcal{A}_s}[r_s^a + \gamma \sum_{s'} p_{ss'}^a V^*(s')]$. Given $V^*$, an optimal policy is easily formed by choosing in each state $s$ any action that achieves the maximum in this equation. A parallel set of value functions, denoted $Q^\pi$ and $Q^*$, and Bellman equations can be defined for state-action pairs, rather than for states. Planning in reinforcement learning refers to the use of models of the environment to compute value functions and thereby to optimize or improve policies.

# 2   Options

We use the term *options* for our generalization of primitive actions to include temporally extended courses of action. Let $h_{t,T} = s_t, a_t, r_{t+1}, s_{t+1}, a_{t+1}, \ldots, r_T, s_T$ be the history sequence from time $t \leq T$ to time $T$, and let $\Omega$ denote the set of all possible histories in the given MDP. Options consist of three components: an initiation set $\mathcal{I} \subseteq \mathcal{S}$, a policy $\pi : \Omega \times \mathcal{A} \to [0, 1]$, and a termination condition $\beta : \Omega \to [0, 1]$. An option $o = \langle \mathcal{I}, \pi, \beta \rangle$ can be taken in state $s$ if and only if $s \in \mathcal{I}$. If $o$ is taken in state $s_t$, the next action $a_t$ is selected according to $\pi(s_t, \cdot)$. The environment then makes a transition to $s_{t+1}$, where $o$ terminates with probability $\beta(h_{t,t+1})$, or else continues, determining $a_{t+1}$ according to $\pi(h_{t,t+1}, \cdot)$, and transitioning to state $s_{t+2}$, where $o$ terminates with probability $\beta(h_{t,t+2})$ etc. We call the general options defined above *semi-Markov* because $\pi$ and $\beta$ depend on the history sequence; in *Markov* options $\pi$ and $\beta$ depend only on the current state. Semi-Markov options allow "timeouts", i.e., termination after some period of time has elapsed, and other extensions which cannot be handled by Markov options.

The initiation set and termination condition of an option together limit the states over which the option's policy must be defined. For example, a hand-crafted policy $\pi$ for a mobile robot to dock with its battery charger might be defined only for states $\mathcal{I}$ in which the battery charger is within sight. The termination condition $\beta$ would be defined to be 1 outside of $\mathcal{I}$ and when the robot is successfully docked.

We can now define *policies over options*. Let the set of options available in state $s$ be denoted $\mathcal{O}_s$; the set of all options is denoted $\mathcal{O} = \bigcup_{s \in \mathcal{S}} \mathcal{O}_s$. When initiated in a state $s_t$, the Markov policy over options $\mu : \mathcal{S} \times \mathcal{O} \to [0, 1]$ selects an option $o \in \mathcal{O}_{s_t}$ according to the probability distribution $\mu(s_t, \cdot)$. The option $o$ is then taken in $s_t$, determining actions until it terminates in $s_{t+k}$, at which point a new option is selected, according to $\mu(s_{t+k}, \cdot)$, and so on. In this way a policy over options, $\mu$, determines a (non-stationary) policy over actions, or *flat policy*, $\pi = f(\mu)$. We define the value of a state $s$ under a general flat policy $\pi$ as the expected return

if the policy is started in $s$:

$$V^\pi(s) \overset{\text{def}}{=} E\left\{ r_{t+1} + \gamma r_{t+2} + \cdots \,\middle|\, \mathcal{E}(\pi, s, t) \right\},$$

where $\mathcal{E}(\pi, s, t)$ denotes the event of $\pi$ being initiated in $s$ at time $t$. The value of a state under a general policy (i.e., a policy over options) $\mu$ can then be defined as the value of the state under the corresponding flat policy: $V^\mu(s) \overset{\text{def}}{=} V^{f(\mu)}(s)$. An analogous definition can be used for the *option-value* function, $Q^\mu(s, o)$. For semi-Markov options it is useful to define $Q^\mu(h, o)$ as the expected discounted future reward after having followed option $o$ through history $h$.

## 3   SMDP Planning

Options are closely related to the actions in a special kind of decision problem known as a *semi-Markov decision process*, or *SMDP* (Puterman, 1994; see also Singh, 1992; Bradtke & Duff, 1995; Mahadevan et. al., 1997; Parr & Russell, 1998). In fact, any MDP with a fixed set of options *is* an SMDP. Accordingly, the theory of SMDPs provides an important basis for a theory of options. In this section, we review the standard SMDP framework for planning, which will provide the basis for our extension.

Planning with options requires a model of their consequences. The form of this model is given by prior work with SMDPs. The reward part of the model of $o$ for state $s \in \mathcal{S}$ is the total reward received along the way:

$$r_s^o = E\left\{ r_{t+1} + \gamma r_{t+2} + \cdots + \gamma^{k-1} r_{t+k} \,\middle|\, \mathcal{E}(o, s, t) \right\},$$

where $\mathcal{E}(o, s, t)$ denotes the event of $o$ being initiated in state $s$ at time $t$. The state-prediction part of the model is

$$p_{ss'}^o = \sum_{k=1}^{\infty} p(s', k) \gamma^k, E\left\{ \gamma^k \delta_{s' s_{t+k}} \mid \mathcal{E}(o, s, t) \right\},$$

for all $s' \in \mathcal{S}$, where $p(s', k)$ is the probability that the option terminates in $s'$ after $k$ steps. We call this kind of model a *multi-time model* because it describes the outcome of an option not at a single time but at potentially many different times, appropriately combined.

Using multi-time models we can write Bellman equations for general policies and options. For any general Markov policy $\mu$, its value functions satisfy the equations:

$$V^\mu(s) = \sum_{o \in \mathcal{O}_s} \mu(s, o) \left[ r_s^o + \sum_{s'} p_{ss'}^o V^\mu(s') \right] \quad \text{and} \quad Q^\mu(s, o) = r_s^o + \sum_{s'} p_{ss'}^o V^\mu(s').$$

Let us denote a restricted set of options by $\mathcal{O}$ and the set of all policies selecting only from options in $\mathcal{O}$ by $\Pi(\mathcal{O})$. Then the optimal value function given that we can select only from $\mathcal{O}$ is $V_\mathcal{O}^*(s) = \max_{o \in \mathcal{O}_s} [r_s^o + \sum_{s'} p_{ss'}^o V_\mathcal{O}^*(s')]$. A corresponding *optimal policy*, denoted $\mu_\mathcal{O}^*$, is any policy that achieves $V_\mathcal{O}^*$, i.e., for which $V^{\mu_\mathcal{O}^*}(s) = V_\mathcal{O}^*(s)$ in all states $s \in \mathcal{S}$. If $V_\mathcal{O}^*$ and the models of the options are known, then $\mu_\mathcal{O}^*$ can be formed by choosing in any proportion among the maximizing options in the equation above for $V_\mathcal{O}^*$.

It is straightforward to extend MDP planning methods to SMDPs. For example, *synchronous value iteration* with options initializes an approximate value function $V_0(s)$ arbitrarily and then updates it by:

$$V_{k+1}(s) \leftarrow \max_{o \in \mathcal{O}_s} [r_s^o + \sum_{s' \in \mathcal{S}} p_{ss'}^o V_k(s')], \quad \forall s \in \mathcal{S}.$$

Note that this algorithm reduces to conventional value iteration in the special case in which $\mathcal{O} = \mathcal{A}$. Standard results from SMDP theory guarantee that such processes converge for

general semi-Markov options: $\lim_{k\to\infty} V_k(s) = V_\mathcal{O}^*(s)$ for all $s \in \mathcal{S}$, $o \in \mathcal{O}$, and for all $\mathcal{O}$. The policies found using temporally abstract options are approximate in the sense that they achieve only $V_\mathcal{O}^*$, which is typically less than the maximum possible, $V^*$.

## 4 Interrupting Options

We are now ready to present the main new insight and result of this paper. SMDP methods apply to options, but only when they are treated as opaque indivisible units. Once an option has been selected, such methods require that its policy be followed until the option terminates. More interesting and potentially more powerful methods are possible by looking inside options and by altering their internal structure (e.g. Sutton, Precup & Singh, 1998).

In particular, suppose we have determined the option-value function $Q^\mu(s, o)$ for some policy $\mu$ and for all state–options pairs $s, o$ that could be encountered while following $\mu$. This function tells us how well we do while following $\mu$ committing irrevocably to each option, but it can also be used to re-evaluate our commitment on each step. Suppose at time $t$ we are in the midst of executing option $o$. If $o$ is Markov in $s$, then we can compare the value of continuing with $o$, which is $Q^\mu(s_t, o)$, to the value of interrupting $o$ and selecting a new option according to $\mu$, which is $V^\mu(s) = \sum_{o'} \mu(s, o') Q^\mu(s, o')$. If the latter is more highly valued, then why not interrupt $o$ and allow the switch? This new way of behaving is indeed better, as shown below.

We can characterize the new way of behaving as following a policy $\mu'$ that is the same as the original one, but over new options, i.e. $\mu'(s, o') = \mu(s, o)$, for all $s \in \mathcal{S}$. Each new option $o'$ is the same as the corresponding old option $o$ except that it terminates whenever switching seems better than continuing according to $Q^\mu$. We call such a $\mu'$ an *interrupted policy* of $\mu$. We will now state a general theorem, which extends the case described above, in that options may be semi-Markov (instead of Markov) and interruption is optional at each state where it could be done. The latter extension lifts the requirement that $Q^\mu$ be completely known, since the interruption can be restricted to states for which this information is available.

**Theorem 1 (Interruption)** *For any MDP, any set of options $\mathcal{O}$, and any Markov policy $\mu : \mathcal{S} \times \mathcal{O} \to [0, 1]$, define a new set of options, $\mathcal{O}'$, with a one-to-one mapping between the two option sets as follows: for every $o = \langle \mathcal{I}, \pi, \beta \rangle \in \mathcal{O}$ we define a corresponding $o' = \langle \mathcal{I}, \pi, \beta' \rangle \in \mathcal{O}'$, where $\beta' = \beta$ except that for any history $h$ in which $Q^\mu(h, o) < V^\mu(s)$, where $s$ is the final state of $h$, we may choose to set $\beta'(h) = 1$. Any histories whose termination conditions are changed in this way are called interrupted histories. Let $\mu'$ be the policy over $o'$ corresponding to $\mu$: $\mu'(s, o') = \mu(s, o)$, where $o$ is the option in $\mathcal{O}$ corresponding to $o'$, for all $s \in \mathcal{S}$. Then*

*1. $V^{\mu'}(s) \geq V^\mu(s)$ for all $s \in \mathcal{S}$.*

*2. If from state $s \in \mathcal{S}$ there is a non-zero probability of encountering an interrupted history upon initiating $\mu'$ in $s$, then $V^{\mu'}(s) > V^\mu(s)$.*

**Proof:** The idea is to show that, for an arbitrary start state $s$, executing the option given by the termination improved policy $\mu'$ and then following policy $\mu$ thereafter is no worse than always following policy $\mu$. In other words, we show that the following inequality holds:

$$\sum_{o'} \mu'(s, o')[r_s^{o'} + \sum_{s'} p_{ss'}^{o'} V^\mu(s')] \geq V^\mu(s) = \sum_o \mu(s, o)[r_s^o + \sum_{s'} p_{ss'}^o V^\mu(s')]. \quad (1)$$

If this is true, then we can use it to expand the left-hand side, repeatedly replacing every occurrence of $V^\mu(x)$ on the left by the corresponding $\sum_{o'} \mu'(x, o')[r_x^{o'} + \sum_{x'} p_{xx'}^{o'} V^\mu(x')]$. In the limit, the left-hand side becomes $V^{\mu'}$, proving that $V^{\mu'} \geq V^\mu$. Since $\mu'(s, o') = \mu(s, o) \; \forall s \in \mathcal{S}$, we need to show that

$$r_s^{o'} + \sum_{s'} p_{ss'}^{o'} V^\mu(s') \geq r_s^o + \sum_{s'} p_{ss'}^o V^\mu(s'). \quad (2)$$

Let $\Gamma$ denote the set of all interrupted histories: $\Gamma = \{h \in \Omega : \beta(h) \neq \beta'(h)\}$. Then, the left hand side of (2) can be re-written as

$$E\left\{r + \gamma^k V^\mu(s') \mid \mathcal{E}(o', s), h_{ss'} \notin \Gamma\right\} + E\left\{r + \gamma^k V^\mu(s') \mid \mathcal{E}(o', s), h_{ss'} \in \Gamma\right\},$$

where $s'$, $r$, and $k$ are the next state, cumulative reward, and number of elapsed steps following option $o$ from $s$ ($h_{ss'}$ is the history from $s$ to $s'$). Trajectories that end because of encountering a history $h_{ss'} \notin \Gamma$ never encounter a history in $\Gamma$, and therefore also occur with the same probability and expected reward upon executing option $o$ in state $s$. Therefore, we can re-write the right hand side of (2) as $E\left\{r + \gamma^k V^\mu(s') \mid \mathcal{E}(o', s), h_{ss'} \notin \Gamma\right\} +$

$$E\left\{\beta(s')[r + \gamma^k V^\mu(s')] + (1 - \beta(s'))[r + \gamma^k Q^\mu(h_{ss'}, o)] \mid \mathcal{E}(o', s), h_{ss'} \in \Gamma\right\}.$$

This proves (1) because for all $h_{ss'} \in \Gamma$, $Q_\mathcal{O}^\mu(h_{ss'}, o) \leq V^\mu(s')$. Note that strict inequality holds in (2) if $Q_\mathcal{O}^\mu(h_{ss'}, o) < V^\mu(s')$ for at least one history $h_{ss'} \in \Gamma$ that ends a trajectory generated by $o'$ with non-zero probability.[1]                                                                            ◇

As one application of this result, consider the case in which $\mu$ is an optimal policy for a given set of Markov options $\mathcal{O}$. The interruption theorem gives us a way of improving over $\mu_\mathcal{O}^*$ with just the cost of checking (on each time step) if a better option exists, which is negligible compared to the combinatorial process of computing $Q_\mathcal{O}^*$ or $V_\mathcal{O}^*$. Kaelbling (1993) and Dietterich (1998) demonstrated a similar performance improvement by interrupting temporally extended actions in a different setting.

## 5   Illustration

Figure 1 shows a simple example of the gain that can be obtained by interrupting options. The task is to navigate from a start location to a goal location within a continuous two-dimensional state space. The actions are movements of length 0.01 in any direction from the current state. Rather than work with these low-level actions, infinite in number, we introduce seven landmark locations in the space. For each landmark we define a controller that takes us to the landmark in a direct path. Each controller is only applicable within a limited range of states, in this case within a certain distance of the corresponding landmark. Each controller then defines an option: the circular region around the controller's landmark is the option's initiation set, the controller itself is the policy, and the arrival at the target landmark is the termination condition. We denote the set of seven landmark options by $\mathcal{O}$. Any action within 0.01 of the goal location transitions to the terminal state, $\gamma = 1$, and the reward is $-1$ on all transitions, which makes this a minimum-time task.

One of the landmarks coincides with the goal, so it is possible to reach the goal while picking only from $\mathcal{O}$. The optimal policy within $\Pi(\mathcal{O})$ runs from landmark to landmark, as shown by the thin line in Figure 1. This is the optimal solution to the SMDP defined by $\mathcal{O}$ and is indeed the best that one can do while picking only from these options. But of course one can do better if the options are not followed all the way to each landmark. The trajectory shown by the thick line in Figure 1 cuts the corners and is shorter. This is the interrupted policy with respect to the SMDP-optimal policy. The interrupted policy takes 474 steps from start to goal which, while not as good as the optimal policy (425 steps), is much better than the SMDP-optimal policy, which takes 600 steps. The state-value functions, $V^{\mu_\mathcal{O}^*}$ and $V^{\mu'}$ for the two policies are also shown in Figure 1.

Figure 2 presents a more complex, mission planning task. A mission is a flight from base to observe as many of a given set of sites as possible and to return to base without running out of fuel. The local weather at each site flips from cloudy to clear according to independent

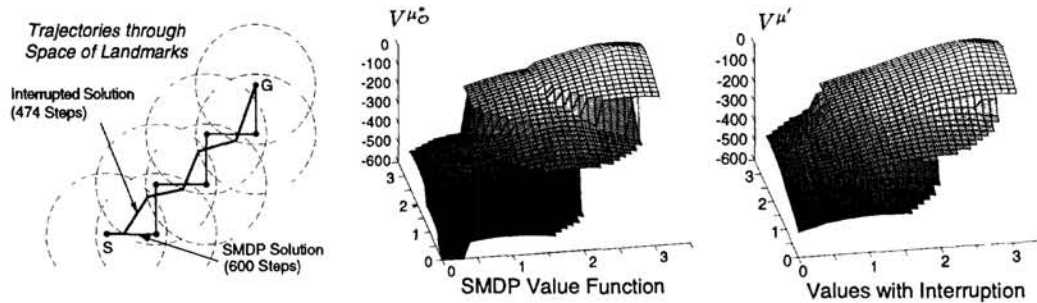

Figure 1: Using interruption to improve navigation with landmark-directed controllers. The task (left) is to navigate from S to G in minimum time using options based on controllers that run each to one of seven landmarks (the black dots). The circles show the region around each landmark within which the controllers operate. The thin line shows the optimal behavior that uses only these controllers run to termination, and the thick line shows the corresponding interrupted behavior, which cuts the corners. The right panels show the state-value functions for the SMDP-optimal and interrupted policies.

Poisson processes. If the sky at a given site is cloudy when the plane gets there, no observation is made and the reward is 0. If the sky is clear, the plane gets a reward, according to the importance of the site. The positions, rewards, and mean time between two weather changes for each site are given in Figure 2. The plane has a limited amount of fuel, and it consumes one unit of fuel during each time tick. If the fuel runs out before reaching the base, the plane crashes and receives a reward of $-100$.

The primitive actions are tiny movements in any direction (there is no inertia). The state of the system is described by several variables: the current position of the plane, the fuel level, the sites that have been observed so far, and the current weather at each of the remaining sites. The state-action space has approximately 24.3 billion elements (assuming 100 discretization levels of the continuous variables) and is intractable by normal dynamic programming methods. We introduced options that can take the plane to each of the sites (including the base), from any position in the state space. The resulting SMDP has only 874,800 elements and it is feasible to exactly determine $V_O^*(s')$ for all sites $s'$. From this solution and the model of the options, we can determine $Q_O^*(s,o) = r_s^o + \sum_{s'} p_{ss'}^o V_O^*(s')$ for any option $o$ and any state $s$ in the whole space.

We performed asynchronous value iteration using the options in order to compute the optimal option-value function, and then used the interruption approach based on the values computed. The policies obtained by both approaches were compared to the results of a static planner, which exhaustively searches for the best tour assuming the weather does not change, and then re-plans whenever the weather does change. The graph in Figure 2 shows the reward obtained by each of these methods, averaged over 100 independent simulated missions. The policy obtained by interruption performs significantly better than the SMDP policy, which in turn is significantly better than the static planner.[2]

## 6   Closing

This paper has developed a natural, even obvious, observation—that one can do better by continually re-evaluating one's commitment to courses of action than one can by committing irrevocably to them. Our contribution has been to formulate this observation precisely enough to prove it and to demonstrate it empirically. Our final example suggests that this technique can be used in applications far too large to be solved at the level of primitive actions. Note that this was achieved using exact methods, without function approximators to represent the value function. With function approximators and other reinforcement learning techniques, it should be possible to address problems that are substantially larger still.

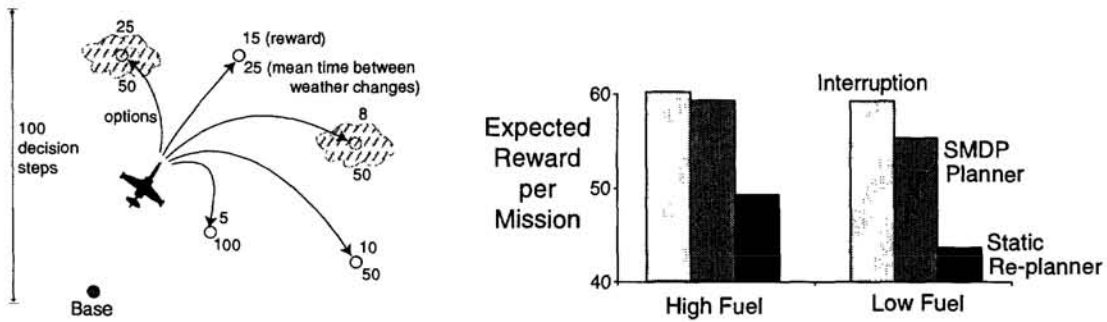

Figure 2: The mission planning task and the performance of policies constructed by SMDP methods, interruption of the SMDP policy, and an optimal static re-planner that does not take into account possible changes in weather conditions.

## Acknowledgments

The authors gratefully acknowledge the substantial help they have received from many colleagues, including especially Amy McGovern, Andrew Barto, Ron Parr, Tom Dietterich, Andrew Fagg, Leo Zelevinsky and Manfred Huber. We also thank Paul Cohen, Robbie Moll, Mance Harmon, Sascha Engelbrecht, and Ted Perkins for helpful reactions and constructive criticism. This work was supported by NSF grant ECS-9511805 and grant AFOSR-F49620-96-1-0254, both to Andrew Barto and Richard Sutton. Satinder Singh was supported by NSF grant IIS-9711753.

## Footnotes

[1]We note that the same proof would also apply for switching to other options (not selected by $\mu$) if they improved over continuing with $o$. That result would be more general and closer to conventional policy improvement. We prefer the result given here because it emphasizes its primary application.

[2]In preliminary experiments, we also used interruption on a crudely learned estimate of $Q_O^*$. The performance of the interrupted solution was very close to the result reported here.

## References

Bradtke, S. J. & Duff, M. O. (1995). Reinforcement learning methods for continuous-time Markov decision problems. In *NIPS 7* (393–500). MIT Press.

Dayan, P. & Hinton, G. E. (1993). Feudal reinforcement learning. In *NIPS 5* (271–278). MIT Press.

Dietterich, T. G. (1998). The MAXQ method for hierarchical reinforcement learning. In *Proceedings of the Fifteenth International Conference on Machine Learning*. Morgan Kaufmann.

Kaelbling, L. P. (1993). Hierarchical learning in stochastic domains: Preliminary results. In *Proceedings of the Tenth International Conference on Machine Learning* (167–173). Morgan Kaufmann.

Mahadevan, S., Marchallek, N., Das, T. K. & Gosavi, A. (1997). Self-improving factory simulation using continuous-time average-reward reinforcement learning. In *Proceedings of the Fourteenth International Conference on Machine Learning* (202–210). Morgan Kaufmann.

McGovern, A., Sutton, R. S., & Fagg, A. H. (1997). Roles of macro-actions in accelerating reinforcement learning. In *Grace Hopper Celebration of Women in Computing* (13–17).

Parr, R. & Russell, S. (1998). Reinforcement learning with hierarchies of machines. In *NIPS 10*. MIT Press.

Puterman, M. L. (1994). *Markov Decision Processes: Discrete Stochastic Dynamic Programming*. Wiley.

Singh, S. P. (1992). Reinforcement learning with a hierarchy of abstract models. In *Proceedings of the Tenth National Conference on Artificial Intelligence* (202–207). MIT/AAAI Press.

Sutton, R. S. (1995). TD models: Modeling the world as a mixture of time scales. In *Proceedings of the Twelfth International Conference on Machine Learning* (531–539). Morgan Kaufmann.

Sutton, R. S., Precup, D. & Singh, S. (1998). Intra-option learning about temporally abstract actions. In *Proceedings of the Fifteenth International Conference on Machine Learning*. Morgan Kaufman.

Sutton, R. S., Precup, D. & Singh, S. (1998). Between MDPs and Semi-MDPs: learning, planning, and representing knowledge at multiple temporal scales. TR 98-74, Department of Comp. Sci., University of Massachusetts, Amherst.

Thrun, S. & Schwartz, A. (1995). Finding structure in reinforcement learning. In *NIPS 7* (385–392). MIT Press.